# At the Edge of Chaos: Real-time Computations and Self-Organized Criticality in Recurrent Neural Networks

**Thomas Natschläger**
Software Competence
Center Hagenberg
A-4232 Hagenberg, Austria
Thomas.Natschlaeger@scch.at

**Nils Bertschinger**
Max Planck Institute for
Mathematics in the Sciences
D-04103 Leipzig, Germany
bertschi@mis.mpg.de

**Robert Legenstein**
Institute for Theoretical
Computer Science, TU Graz
A-8010 Graz, Austria
legi@igi.tu-graz.ac.at

## Abstract

In this paper we analyze the relationship between the computational capabilities of randomly connected networks of threshold gates in the time-series domain and their dynamical properties. In particular we propose a complexity measure which we find to assume its highest values near the edge of chaos, i.e. the transition from ordered to chaotic dynamics. Furthermore we show that the proposed complexity measure predicts the computational capabilities very well: only near the edge of chaos are such networks able to perform complex computations on time series. Additionally a simple synaptic scaling rule for self-organized criticality is presented and analyzed.

## 1 Introduction

It has been proposed that extensive computational capabilities are achieved by systems whose dynamics is neither chaotic nor ordered but somewhere in between order and chaos. This has led to the idea of "*computation at the edge of chaos*". Early evidence for this hypothesis has been reported e.g. in [1]. The results of numerous computer simulations carried out in these studies suggested that there is a sharp transition between ordered and chaotic dynamics. Later on this was confirmed by Derrida and others [2]. They used ideas from statistical physics to develop an accurate mean-field theory which allowed to determine the critical parameters analytically. Because of the physical background, this theory focused on the autonomous dynamics of the system, i.e. its relaxation from an initial state (the input) to some terminal state (the output) without any external influences. In contrast to such "off-line" computations, we will focus in this article on time-series computations, i.e. mappings, also called filters, from a time-varying input signal to a time-varying output signal. Such "online" or real-time computations describe more adequately the input to output relation of systems like animals or autonomous robots which must react in real-time to a continuously changing stream of sensory input.

The purpose of this paper is to analyze how the computational capabilities of randomly connected recurrent neural networks in the domain of real-time processing and the type of dynamics induced by the underlying distribution of synaptic weights are related to each other. In particular, we will show that for the types of neural networks considered in this paper (defined in Sec. 2) there also exists a transition from ordered to chaotic dynamics. This phase transition is determined using an extension of the mean-field approach described in [3] and [4] (Sec. 3). As the next step we propose a novel complexity measure (Sec. 4) which

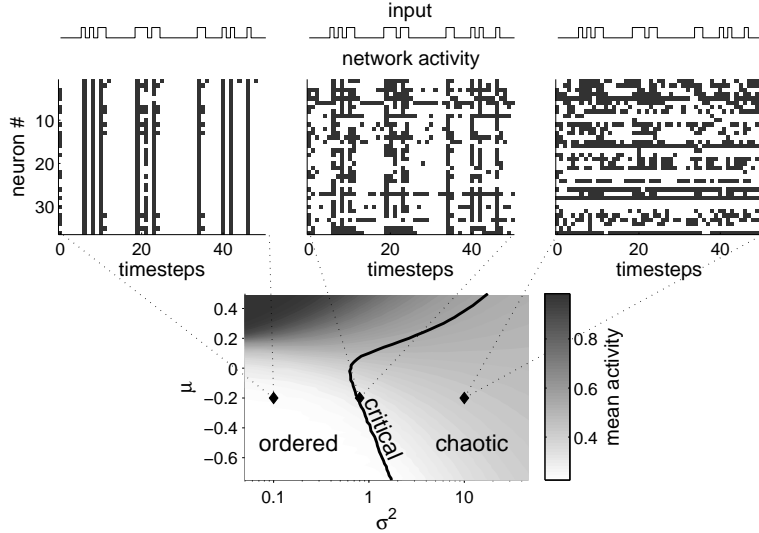

Figure 1: Networks of randomly connected threshold gates can exhibit ordered, critical and chaotic dynamics. In the upper row examples of the temporal evolution of the network state $\boldsymbol{x}_t$ are shown (black: $x_{i,t} = 1$, white: $x_{i,t} = 0$, input as indicated above) for three different networks with parameters taken from the ordered, critical and chaotic regime, respectively. Parameters: $K = 5$, $N = 500$, $\bar{u} = -0.5$, $r = 0.3$ and $\mu$ and $\sigma^2$ as indicated in the phase plot below. The background of the phase plot shows the mean activity $a^*$ (see Sec. 3) of the networks depending on the parameters $\mu$ and $\sigma^2$.

can be calculated using the mean-field theory developed in Sec. 3 and serves as a predictor for the computational capability of a network in the time-series domain. Employing a recently developed framework for analyzing real-time computations [5, 6] we investigate in Sec. 5 the relationship between network dynamics and the computational capabilities in the time-series domain. In Sec. 6 of this paper we propose and analyze a synaptic scaling rule for self-organized criticality (SOC) for the types of networks considered here. In contrast to previous work [7], we do not only check that the proposed rule shows adaptation towards critical dynamics, but also show that the computational capabilities of the network are actually increased if the rule is applied.

*Relation to previous work:* In [5], the so-called *liquid state machine* (LSM) approach was proposed and used do analyze the computational capabilities in the time-series domain of randomly connected networks of biologically inspired network models (composed of leaky integrate-and-fire neurons). We will use that approach to demonstrate that only near the edge of chaos, complex computations can be performed (see Sec. 5). A similar analysis for a restricted case (zero mean of synaptic weights) of the network model considered in this paper can be found in [4].

## 2   The Network Model and its Dynamics

We consider *input driven* recurrent networks consisting of $N$ threshold gates with states $x_i \in \{0, 1\}$. Each node $i$ receives nonzero incoming weights $w_{ij}$ from exactly $K$ randomly chosen nodes $j$. Each nonzero connection weight $w_{ij}$ is randomly drawn from a Gaussian distribution with mean $\mu$ and variance $\sigma^2$. Furthermore, the network is *driven by an external input signal* $u_{(.)}$ which is injected into each node. Hence, in summary, the update of the network state $\boldsymbol{x}_t = (x_{1,t}, \ldots, x_{N,t})$ is given by $x_{i,t} = \Theta(\sum_{j=1}^{N} w_{ij} \cdot x_{j,t-1} + u_{t-1})$ which is applied to all neurons in parallel and where $\Theta(h) = 1$ if $h \geq 0$ and $\Theta(h) = 0$ otherwise. In the following we consider a randomly drawn binary input signal $u_{(.)}$: at each

time step $u_t$ assumes the value $\bar{u} + 1$ with probability $r$ and the value $\bar{u}$ with probability $1 - r$. This network model is similar to the one we have considered in [4]. However it differs in two important aspects: a) By using states $x_i \in \{0, 1\}$ we emphasis the asymmetric information encoding by spikes prevalent in biological neural systems and b) it is more general in the sense that the Gaussian distribution from which the non-zero weights are drawn is allowed to have an arbitrary mean $\mu \in \mathbb{R}$. This implies that the network activity $a_t = \frac{1}{N} \sum_{i=1}^{N} x_{i,t}$ can vary considerably for different parameters (compare Fig. 1) and enters all the calculations discussed in the rest of the paper.

The top row of Fig. 1 shows typical examples of ordered, critical and chaotic dynamics (see the next section for a definition of order and chaos). The system parameters corresponding to each type of dynamics are indicated in the lower panel (*phase plot*). We refer to the (phase) transition from the ordered to the chaotic regime as the *critical line* (shown as the solid line in the phase plot). Note that increasing the variance $\sigma^2$ of the weights consistently leads to chaotic behavior.

## 3   The Critical Line: Order and Fading Memory versus Chaos

To define the chaotic and ordered phase of an *input driven* network we use an approach which is similar to that proposed by Derrida and Pomeau [2] for autonomous systems: consider two (initial) network states with a certain (normalized) Hamming distance. These states are mapped to their corresponding successor states (using the same weight matrix) *with the same input* in each case and the change in the Hamming distance is observed. If small distances tend to grow this is a sign of chaos whereas if the distance tends to decrease this is a signature of order.

Following closely the arguments in [4, 3] we developed a mean-field theory (see [8] for all details) which allows to calculate the update $d_{t+1} = f(d_t, a_t, u_t)$ of the normalized Hamming distance $d_t = |\{i : x_{i,t} \neq \tilde{x}_{i,t}\}|/N$ between two states $\boldsymbol{x}_t$ and $\tilde{\boldsymbol{x}}_t$ as well as the update $a_{t+1} = A(a_t, u_t)$ of the network activity in one time step. Note that $d_{t+1}$ depends on the input $u_t$ (in contrast to [3]) and also on the activity $a_t$ (in contrast to [4]). Hence the two-dimensional map $F_u(d_t, a_t) := (d_{t+1}, a_{t+1}) = (f(d_t, a_t, u_t), A(a_t, u_t))$ describes the time evolution of $d_t$ and $a_t$ given the input times series $u_{(\cdot)}$.

Let us consider the steady state of the averaged Hamming distance $f^*$ as well as the steady state of the averaged network activity $a^*$, i.e. $(f^*, a^*) = \lim_{t \to \infty} \langle F_u^t \rangle$.[1] If $f^* = 0$ we know that any state differences will eventually die out and the network is in the ordered phase. If on the other hand a small difference is amplified and never dies out we have $f^* \neq 0$ and the network is in the chaotic phase. Whether $f^* = 0$ or $f^* \neq 0$ can be decided by looking at the slope of the function $f(\cdot, \cdot, \cdot)$ at its fixed point $f^* = 0$. Since $a_t$ does not depend on $d_t$ we calculate the averaged steady state activity $a^*$ and determine the slope $\alpha^*$ of the map $r f(d, a, \bar{u} + 1) + (1 - r) f(d, a, \bar{u})$ at the point $(d, a) = (0, a^*)$. Accordingly we say that the network is in the ordered, critical or chaotic regime if $\alpha^* < 1$, $\alpha^* = 1$ or $\alpha^* > 1$ respectively. In [8] it is shown that the so called critical line $\alpha^* = 1$ where the phase transition from ordered to chaotic behavior occurs is given by

$$P_{bf} = \sum_{n=0}^{K-1} \binom{K-1}{n} a^{*n} (1 - a^*)^{K-1-n} (r Q(1, n, \bar{u} + 1) + (1 - r) Q(1, n, \bar{u})) = \frac{1}{K} \quad (1)$$

where $P_{bf}$ denotes the probability (averaged over the inputs and the network activity) that a node will change its output if a single out of its $K$ input bits is flipped.[2]  Examples of

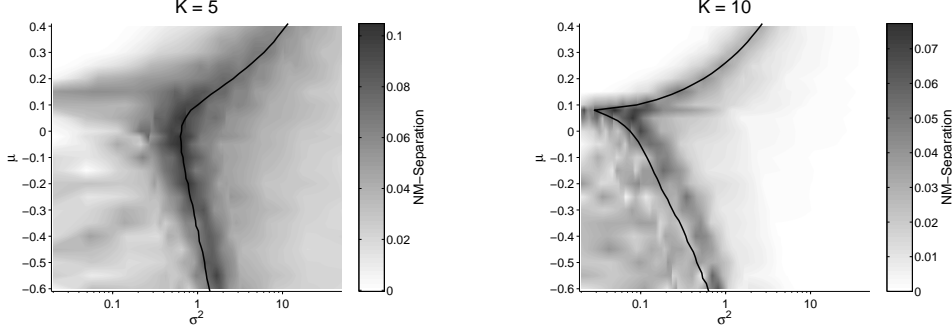

Figure 2: $NM$-separation assumes high values on the critical line. The gray coded image shows the $NM$-separation in dependence on $\mu$ and $\sigma^2$ for $K$ denoted in the panels, $r = 0.3$, $\bar{u} = -0.5$ and $b = 0.1$. The solid line marks the critical values for $\mu$ and $\sigma^2$.

critical lines that were calculated from this formula (marked by the solid lines) can be seen in Fig. 2 for $K = 5$ and $K = 10$.[3]

The ordered phase can also be described by using the notion of *fading memory* (see [5] and the references therein). Intuitively speaking in a network with fading memory a state $x_t$ is fully determined by a finite history $u_{t-T}, u_{t-T+1}, \dots, u_{t-1}, u_t$ of the input $u_{(\cdot)}$. A slight reformulation of this property (see [6] and the references therein) shows that it is equivalent to the requirement that all state differences vanish, i.e. being in the ordered phase. Fading memory plays an important role in the "liquid state machine" framework [5] since together with the separation property (see below) it would in principle allow an appropriate *readout function* to deduce the recent input, or any function of it, from the network state. If on the other hand the network does not have fading memory (i.e. is in the chaotic regime) a given network state $x_t$ also contains "spurious" information about the initial conditions and hence it is hard or even impossible to deduce any features of the recent input.

## 4  *NM*-Separation as a Predictor for Computational Power

The already mentioned *separation property* [5] is especially important if a network is to be useful for computations on input time-series: only if different input signals separate the network state, i.e. different inputs result in different states, it is possible for a readout function to respond differently. Hence it is necessary that any two different input time series for which the readout function should produce different outputs drive the recurrent network into two sufficiently different states.

The mean field theory we have developed (see [8]) can be extended to describe the update $d_{t+1} = s(d_t, \dots)$ of the Hamming distance that result from applying different inputs $u_{(\cdot)}$ and $\tilde{u}_{(\cdot)}$ with a mean distance of $b := \Pr\{u_t \neq \tilde{u}_t\}$, i.e. the separation. In summary the three-dimensional map $S_{u,\tilde{u}}(d_t, a_t, \tilde{a}_t) := (d_{t+1}, a_{t+1}, \tilde{a}_{t+1}) = (s(d_t, a_t, \tilde{a}_t, u_t, \tilde{u}_t), A(a_t, u_t), A(\tilde{a}_t, \tilde{u}_t))$ fully describes the time evolution of the Hamming distance and the network activities. Again we consider the steady state of the averaged Hamming distance $s^*$ and the network activities $a^*, \tilde{a}^*$, i.e. $(s^*, a^*, \tilde{a}^*) = \lim_{t \to \infty} \langle S_{u,\tilde{u}}^t \rangle$.

The overall separation for a given input statistics (determined by $\bar{u}$, $r$, and $b$) is then given by $s^*$. However, this overall separation measure can not be directly related to the computa-

---

external input $u$ and is given by $Q(1, n, u) = \int_{-\infty}^{-u} \phi(\xi, n\mu, n\sigma^2) \left(1 - \Phi(-u - \xi, \mu, \sigma^2)\right) d\xi + \int_{-u}^{\infty} \phi(\xi, n\mu, n\sigma^2) \Phi(-u - \xi, \mu, \sigma^2) d\xi$ where $\phi$, $\Phi$ denote the Gaussian density and cumulative density respectively (see [8] for a detailed explanation).

[3]For each value of $\mu = -0.6 + k * 0.01$, $k = 0 \dots 100$ a search was conducted to find the value for $\sigma^2$ such that $\alpha^* = 1$. Numerical iterations of the function $A$ were used to determine $a^*$.

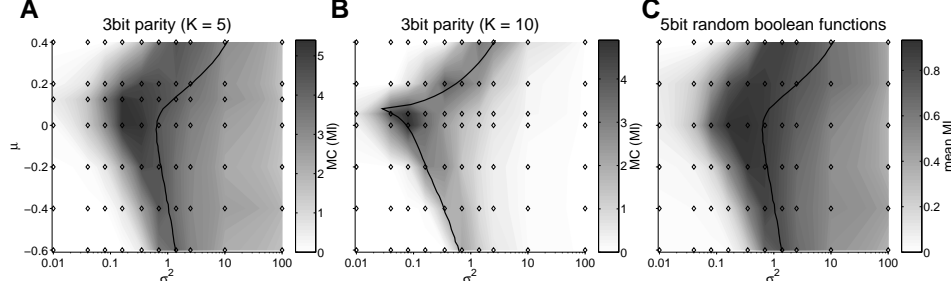

Figure 3: Real-time computation at the edge of chaos. **A** The gray coded image (an interpolation between the data points marked with open diamonds) shows the performance of trained networks in dependence of the parameters $\mu$ and $\sigma^2$ for the delayed 3-bit parity task. Performance is measured as the memory capacity $MC = \sum_\tau I(v, y^{(\tau)})$ where $I(v, y^{(\tau)})$ is the mutual information between the classifier output $v_{(\cdot)}$ and the target function $y_t^{(\tau)} = \text{PARITY}(u_{t-\tau}, u_{t-\tau-1}, u_{t-\tau-2})$ measured on a test set. **B** Same as panel A but for $K = 10$. **C** Same as panel A but for an average over 50 randomly drawn Boolean functions $f$ of the last 5 time steps, i.e. $y_t = f(u_t, u_{t-1}, ..., u_{t-4})$.

tional power since chaotic networks separate even minor differences in the input to a very high degree. The part of this separation that is caused by the input distance $b$ and not by the distance of some initial state is therefore given by $s^* - f^*$ because $f^*$ measures the state distance that is caused by differences in the initial states and remains even after long runs with the same inputs (see Sec. 3). Note that $f^*$ is always zero in the ordered phase and non-zero in the chaotic phase.

Since we want the complexity measure, which we will call $NM$-separation, to be a predictor for computational power we correct $s^* - f^*$ by a term which accounts for the separation due to an all-dominant input drive. A suitable measure for this "immediate separation" $i^*$ is the average *increase* in the Hamming distance if the system is run for a long time ($t \to \infty$) with equal inputs $u_{(\cdot)} = \tilde{u}_{(\cdot)}$ and then a single step with an input pair $(v, \tilde{v})$ with an average difference of $b = \Pr\{v, \neq \tilde{v}\}$ is applied: $i^* = \lim_{t\to\infty} \sum_{v,\tilde{v}=0}^{1} r^v (1-r)^{1-v} b^{|v-\tilde{v}|} (1-b)^{1-|v-\tilde{v}|} \langle s(\cdot, \cdot, \cdot, v, \tilde{v}) \circ S_{u,u}^t \rangle - f^*$. Hence a measure of the network mediated separation $NM_{sep}$ due to input differences is given by

$$NM_{sep} = s^* - f^* - i^* \qquad (2)$$

In Fig. 2 the $NM$-separation resulting from an input difference of $b = 0.1$ is shown in dependence of the network parameters $\mu$ and $\sigma^2$.[4] Note that the $NM$-separation peaks very close to the critical line. Because of the computational importance of the separation property this also suggests that the computational capabilities of the networks will peak at the onset of chaos, which is confirmed in the next section.

## 5 Real-Time Computations at the Edge of Chaos

To access the computational power of a network we make use of the so called "liquid state machine" framework which was proposed by Maass et.al. [5] and independently by Jaeger [6]. They put forward the idea that any complex time-series computation can be implemented by composing a system which consists of two conceptually different parts: a) a

properly chosen general-purpose recurrent network with "rich" dynamics and b) a read-out function that is trained to map the network state to the desired outputs (see [5, 6, 4] for more details). This approach is potentially successful if the general-purpose network encodes the relevant features of the input signal in the network state in such a way that the readout function can easily extract it. We will show that near the critical line the networks considered in this paper encode the input in such a way that a simple linear classifier $C(\boldsymbol{x}_t) = \Theta(\boldsymbol{w} \cdot \boldsymbol{x}_t + w_0)$ suffices to implement a broad range of complex nonlinear filters. Note that in order to train the network for a given task only the parameters $\boldsymbol{w} \in \mathbb{R}^N$, $w_0 \in \mathbb{R}$ of the linear classifier are adjusted such that the actual network output $v_t = C(\boldsymbol{x}_t)$ is as close as possible to the target values $y_t$.

To access the computational power in a principled way networks with different parameters were tested on a delayed 3-bit parity task for increasing delays and on randomly drawn Boolean functions of the last 5 input bits. Note that these tasks are quite complex for the networks considered here since most of them are not linear separable (i.e. the parity function) and require memory. Hence to achieve good performance it is necessary that a state $\boldsymbol{x}_t$ contains information about several input bits $u_{t'}$, $t' < t$ in a nonlinear transformed form such that a linear classifier $C$ is sufficient to perform the nonlinear computations.

The results are summarized in Fig. 3 where the performance (measured in terms of mutual information) on a test set between the network output and the target signal is shown for various parameter settings (for details see [4]). The highest performance is clearly achieved for parameter values close to the critical line where the phase transition occurs. This has been noted before [1]. In contrast to these previous results the networks used here are not optimized for any specific task but their computational capabilities are assessed by evaluating them for many different tasks. Therefore a network that is specifically designed for a single task will not show a good performance in this setup. These considerations suggest the following hypotheses regarding the computational function of generic recurrent neural circuits: to serve as a general-purpose temporal integrator, and simultaneously as a kernel (i.e., nonlinear projection into a higher dimensional space) to facilitate subsequent (linear) readout of information whenever it is needed.

## 6  Self-Organized Criticality via Synaptic Scaling

Since the computational capabilities of a network depend crucially on having almost critical dynamics an adaptive system should be able to adjust its dynamics accordingly.

Equ. (1) states that critical dynamics are achieved if the probability $P_{bf}$ that a single bit-flip in the input shows up in the output should on average (over the external and internal input statistics given by $\bar{u}$, $r$ and $a^*$ respectively) be equal to $\frac{1}{K}$. To allow for a rule that can adjust the weights of each node a local estimate of $P_{bf}$ must be available. This can be accomplished by estimating $P_{bf}$ from the margin of each node, i.e. the distance of the internal activation from the firing threshold. Intuitively a node with an activation that is much higher or lower than its firing threshold is rather unlikely to change its output if a single bit in its input is flipped. Formally $P_{bf}^i$ of node $i$ is given by the average (over the internal and external input statistics) of the following quantity:

$$\frac{1}{K} \sum_{j=1, w_{ij} \neq 0}^{N} \Theta\left(w_{ij}(1 - 2x_{j,t-1})(1 - 2x_{i,t}) - m_{i,t}\right) \tag{3}$$

where $m_{i,t} = \left|\sum_{j=1}^{N} w_{ij}x_{j,t-1} + u_{t-1}\right|$ denotes the margin of node $i$ (see [8] for details). Each node now applies synaptic scaling to adjust itself towards the critical line. Accordingly we arrive at the following SOC-rule:

$$w_{ij}(t+1) = \begin{cases} \frac{1}{1+\nu} \cdot w_{ij} & \text{if } P_{bf}^{est_i}(t) > \frac{1}{K} \\ (1+\nu) \cdot w_{ij}(t) & \text{if } P_{bf}^{est_i}(t) < \frac{1}{K} \end{cases} \tag{4}$$

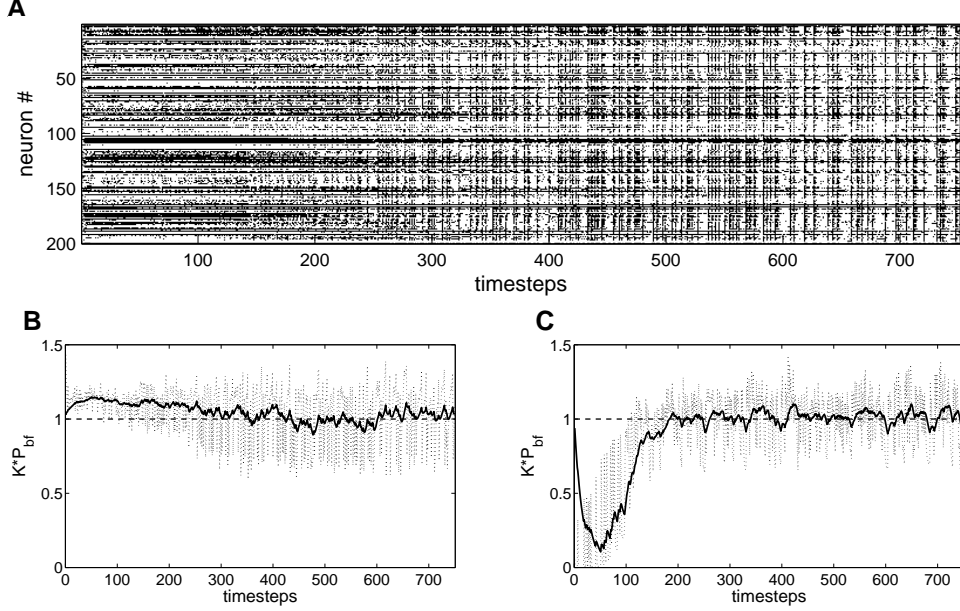

Figure 4: Self-organized criticality. **A** Time evolution of the network state $x_t$ starting in a chaotic regime while the SOC-rule (4) is active (black: $x_{i,t} = 1$, white: $x_{i,t} = 0$). Parameters: $N = 500, K = 5, \bar{u} = -0.5, r = 0.3, \mu = 0$ and initial $\sigma^2 = 100$. **B** Estimated $P_{bf}$. The dotted line shows how the node averaged estimate of $P_{bf}$ evolves over time for the network shown in A. The running average of this estimate (thick black line) as used by the SOC-rule clearly shows that $P_{bf}$ approaches its critical value (dashed line). **C** Same as B but for $K = 10$ and initial $\sigma^2 = 0.01$ in the ordered regime.

where $0 < \nu \ll 1$ is the learning rate and $P_{bf}^{est_i}(t)$ is a running average of the formula in Equ. (3) to estimate $P_{bf}^i$. Applying this rule in parallel to all nodes of the network is then able to adjust the network dynamics towards criticality as shown in Fig. 4[5]. The upper row shows the time evolution of the network states $x_t$ while the SOC-rule (4) is running. It is clearly visible how the network dynamics changes from chaotic (the initial network had the parameters $K = 5, \mu = 0$ and $\sigma^2 = 100$) to critical dynamics that respect the input signal. The lower row of Fig. 4 shows how the averaged estimated bit-flip probability $\frac{1}{N} \sum_{i=1}^{N} P_{bf}^{est_i}(t)$ approaches its critical value for the case of the above network and one that started in the ordered regime ($K = 10, \mu = 0, \sigma^2 = 0.01$).

Since critical dynamics are better suited for information processing (see Fig. 3) it is expected that the performance on the 3-bit parity task improves due to SOC. This is confirmed in Fig. 5 which shows how the memory capacity $MC$ (defined in Fig. 3) grows for networks that were initialized in the chaotic and ordered regime respectively. Note that the performance reached by these networks using the SOC-rule (4) is as high as for networks where the critical value for $\sigma^2$ is chosen apriori and stays at this level. This shows that rule (4) is stable in the sense that it keeps the dynamics critical and does not destroy the computational capabilities.

## 7 Discussion

We developed a mean-field theory for input-driven networks which allows to determine the position of the transition line between ordered and chaotic dynamics with respect to the

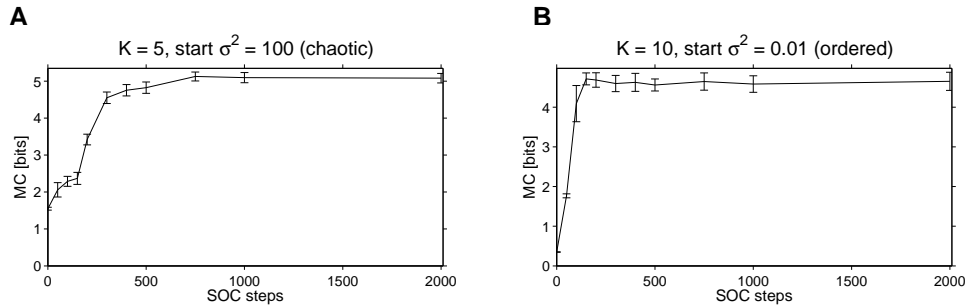

Figure 5: Time evolution of the performance with activated SOC-rule. **A** The plot shows the memory capacity $MC$ (see Fig. 3) on the 3-bit parity task averaged over 25 networks ($\pm$ standard deviation as error-bars) evaluated at the indicated time steps. At each evaluation time step the network weights were fixed and the $MC$ was measured as in Fig. 3 by training the corresponding readouts from scratch. The networks were initialized in the chaotic regime. **B** Same as in A but for $K = 10$ and networks initialized in the ordered regime.

parameters controlling the network connectivity and input statistics. Based on this theory we proposed a complexity measure (called $NM$-separation) which assumes its highest values at the critical line and shows a clear correlation with the computational power for real-time time-series processing. These results provide further evidence for the idea of "computation at the edge of chaos" [1] and support the hypothesis that dynamics near the critical line are expected to be a general property of input driven dynamical systems which support complex real-time computations. Therefore our analysis and the proposed complexity measure provide a new approach towards discovering dynamical principles that enable biological systems to do sophisticated information processing.

Furthermore we have shown that a local rule for synaptic scaling is able to adjust the weights of a network towards critical dynamics. Additionally networks adjusting themselves by this rule have been found to exhibit enhanced computational capabilities. Thereby systems can combine task-specific optimization provided by (supervised) learning rules with self-organization of its dynamics towards criticality. This provides an explanation how specific information can be processed while still being able to react to incoming signals in a flexible way.

**Acknowledgement**   This work was supported in part by the PASCAL project #IST-2002-506778 of the European Community.

## Footnotes

[1] $F_u^t$ denotes $t$-fold composition of the map $F_u(\cdot, \cdot)$ where in the $k$-th iteration the input $u_k$ is applied and $\langle \cdot \rangle$ denotes the average over all possible initial conditions and all input signals with a given statistics determined by $\bar{u}$ and $r$.

[2] The actual single bit-flip probability $Q$ depends on the number $n$ of inputs which are 1 and the

[4]For each value of $\mu = -0.6 + k * 0.05$, $k = 0 \dots 20$, 10 values for $\sigma^2$ where chosen near the critical line and 10 other values where equally spaced (on a logarithmic scale) over the interval [0.02,50]. For each such pair $(\mu, \sigma^2)$ extensive numerical iterations of the map $S$ where performed to obtain accurate estimates of $s^*, f^*$ and $i^*$. Hopefully these numerical estimates can be replaced by analytic results in the future.

[5]Here a learning rate of $\nu = 0.01$ and an exponentially weighted running average with a time constant of 15 time steps were used.

## References

[1] C. G. Langton. Computation at the edge of chaos. *Physica D*, 42, 1990.

[2] B. Derrida and Y. Pomeau. Random networks of automata: A simple annealed approximation. *Europhys. Lett.*, 1:45–52, 1986.

[3] B. Derrida. Dynamical phase transition in non-symmetric spin glasses. *J. Phys. A: Math. Gen.*, 20:721–725, 1987.

[4] N. Bertschinger and T. Natschläger. Real-time computation at the edge of chaos in recurrent neural networks. *Neural Computation*, 16(7):1413–1436, 2004.

[5] W. Maass, T. Natschläger, and H. Markram. Real-time computing without stable states: A new framework for neural computation based on perturbations. *Neural Computation*, 14(11), 2002.

[6] H. Jaeger and H. Haas. Harnessing nonlinearity: Predicting chaotic systems and saving energy in wireless communication. *Science*, 304(5667):78–80, 2004.

[7] S. Bornholdt and T. Röhl. Self-organized critical neural networks. *Physical Review E*, 67:066118, 2003.

[8] N. Bertschinger and T. Natschläger. Supplementary information to the mean-field theory for randomly connected recurrent networks of threshold gates, 2004. http://www.igi.tugraz.at/tnatschl/edge-of-chaos/mean-field-supplement.pdf.